# Spike-Timing-Dependent Learning for Oscillatory Networks

**Silvia Scarpetta**
Dept. of Physics "E.R. Caianiello"
Salerno University 84081 (SA) Italy
and INFM, Sezione di Salerno Italy
*scarpetta@na.infn.it*

**Zhaoping Li**
Gatsby Comp. Neurosci. Unit
University College, London, WC1N 3AR
United Kingdom
*zhaoping@gatsby.ucl.ac.uk*

**John Hertz**
Nordita
2100 Copenhagen Ø, Denmark
*hertz@nordita.dk*

## Abstract

We apply to oscillatory networks a class of learning rules in which synaptic weights change proportional to pre- and post-synaptic activity, with a kernel $A(\tau)$ measuring the effect for a postsynaptic spike a time $\tau$ after the presynaptic one. The resulting synaptic matrices have an outer-product form in which the oscillating patterns are represented as complex vectors. In a simple model, the even part of $A(\tau)$ enhances the resonant response to learned stimulus by reducing the effective damping, while the odd part determines the frequency of oscillation. We relate our model to the olfactory cortex and hippocampus and their presumed roles in forming associative memories and input representations.

## 1 Introduction

Recent studies of synapses between pyramidal neocortical and hippocampal neurons [1, 2, 3, 4] have revealed that changes in synaptic efficacy can depend on the relative timing of pre- and postsynaptic spikes. Typically, a presynaptic spike followed by a postsynaptic one leads to an increase in efficacy (LTP), while the reverse temporal order leads to a decrease (LTD). The dependence of the change in synaptic efficacy on the difference $\tau$ between the two spike times may be characterized by a kernel which we denote $A(\tau)$ [4]. For hippocampal pyramidal neurons, the half-width of this kernel is around 20 ms.

Many important neural structures, notably hippocampus and olfactory cortex, exhibit oscillatory activity in the 20-50 Hz range. Here the temporal variation of the neuronal firing can clearly affect the synaptic dynamics, and vice versa. In this paper we study a simple model for learning oscillatory patterns, based on the structure of the kernel $A(\tau)$ and other known physiology of these areas. We will assume

that these synaptic changes in long range lateral connections are driven by oscillatory, patterned input to a network that initially has only local synaptic connections. The result is an imprinting of the oscillatory patterns in the synapses, such that subsequent input of a similar pattern will evoke a strong resonant response. It can be viewed as a generalization to oscillatory networks with spike-timing-dependent learning of the standard scenario whereby stationary patterns are stored in Hopfield networks using the conventional Hebb rule.

## 2 Model

The computational neurons of the model represent local populations of biological neurons that share common input. They follow the equations of motion [5]

$$\dot{u}_i = -\alpha u_i - \beta_i^0 g_v(v_i) + \sum_j J_{ij}^0 g_u(u_j) + I_i, \tag{1}$$

$$\dot{v}_i = -\alpha v_i + \gamma_i^0 g_u(u_i) + \sum_{j \neq i} W_{ij}^0 g_u(u_j). \tag{2}$$

Here $u_i$ and $v_i$ are membrane potentials for excitatory and inhibitory (formal) neuron $i$, $\alpha^{-1}$ is their membrane time constant, and the sigmoidal functions $g_u(\ )$ and $g_v(\ )$ model the dependence of their outputs (interpreted as instantaneous firing rates) on their membrane potentials. The couplings $\beta_i^0$ and $\gamma_i^0$ are inhibitory-to-excitatory (resp. excitatory-to-inhibitory) connection strengths within local excitatory-inhibitory pairs, and for simplicity we take the external drive $I_i(t)$ to act only on the excitatory units. We include nonlocal excitatory couplings $J_{ij}^0$ between excitatory units and $W_{ij}^0$ from excitatory units to inhibitory ones. In this minimal model, we ignore long-range inhibitory couplings, appealing to the fact that real anatomical inhibitory connections are predominantly short-ranged. (In what follows, we will sometimes use bold and sans serif notation (e.g., $\mathbf{u}$, $\mathsf{J}$) for vectors and matrices, respectively.) The structure of the couplings is shown in Fig. 1A.

The model is nonlinear, but here we will limit our treatment to an analysis of small oscillations around a stable fixed point $\{\bar{\mathbf{u}}, \bar{\mathbf{v}}\}$ determined by the DC part of the input. Performing the linearization and eliminating the inhibitory units [6, 5], we obtain

$$\ddot{\mathbf{u}} + [2\alpha - \mathsf{J}]\dot{\mathbf{u}} + [\alpha^2 + \beta(\gamma + \mathsf{W}) - \alpha \mathsf{J}]\mathbf{u} = (\partial_t + \alpha)\delta \mathbf{I}. \tag{3}$$

Here $\mathbf{u}$ is now measured from the fixed point $\bar{\mathbf{u}}$, $\delta \mathbf{I}$ is the time-varying part of the input, and the elements of $\mathsf{J}$ and $\mathsf{W}$ are related to those of $\mathsf{J}^0$ and $\mathsf{W}^0$ by $W_{ij} = g_u'(\bar{u}_j)W_{ij}^0$ and $J_{ij} = g_u'(\bar{u}_j)J_{ij}^0$. For simplicity, we have assumed that the effective local couplings $\beta_i = g_v'(\bar{v}_i)\beta_i^0$ and $\gamma_i = g_u'(\bar{u}_i)\gamma_i^0$ are independent of $i$: $\beta_i = \beta$, $\gamma_i = \gamma$. With oscillatory inputs $\delta \mathbf{I} = \boldsymbol{\xi} e^{-i\omega t} +$ c.c., the oscillatory pattern elements $\xi_i = |\xi_i| e^{-i\phi_i}$ are complex, reflecting possible phase differences across the units. We likewise separate the response $\mathbf{u} = \mathbf{u}^+ + \mathbf{u}^-$ (after the initial transients) into positive- and negative-frequency components $\mathbf{u}^\pm$ (with $\mathbf{u}^- = \mathbf{u}^{+*}$ and $\mathbf{u}^\pm \propto e^{\mp i\omega t}$). Since $\dot{\mathbf{u}}^\pm = \mp i\omega \mathbf{u}^\pm$, Eqn. (3) can be written

$$\left[2\alpha \pm \frac{i}{\omega}(\alpha^2 + \beta\gamma - \omega^2)\right]\mathbf{u}^\pm = \mathsf{M}^\pm \mathbf{u}^\pm + \left(1 \pm \frac{i\alpha}{\omega}\right)\delta \mathbf{I}^\pm, \tag{4}$$

a form that shows how the matrix

$$\mathsf{M}^\pm(\omega) \equiv \mathsf{J} \mp \frac{i}{\omega}(\beta \mathsf{W} - \alpha \mathsf{J}). \tag{5}$$

describes the effective coupling between local oscillators. $2\alpha$ is the intrinsic damping and $\sqrt{\alpha^2 + \beta\gamma}$ the frequency of the individual oscillators.

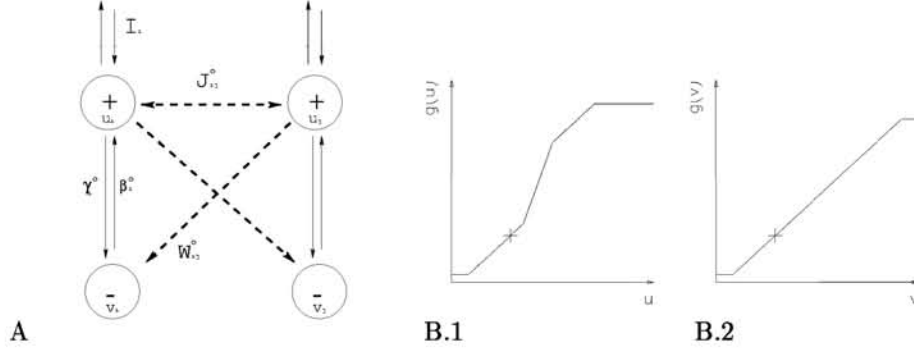

A                          B.1             B.2

Figure 1: A. The model: In addition to the local excitatory-inhibitory connections (vertical solid lines), there are nonlocal long-range connections (dashed lines) between excitatory units ($J_{ij}$) and from excitatory to inhibitory units ($W_{ij}$). External inputs are fed to the excitatory units. B: Activation function used in simulations for excitatory units (B.1) and inhibitory units (B.2). Crosses mark the equilibrium point ($\bar{u}, \bar{v}$) of the system.

## 2.1   Learning phase

We employ a generalized Hebb rule of the form

$$\delta C_{ij}(t) = \eta \int_0^T dt \int_{-\infty}^{\infty} d\tau \, y_i(t+\tau) A(\tau) x_j(t) \qquad (6)$$

for synaptic weight $C_{ij}$, where $x_j$ and $y_i$ are the pre- and postsynaptic activities, measured relative to stationary levels at which no changes in synaptic strength occur. We consider a general kernel $A(\tau)$, although experimentally $A(\tau) > 0$ ($< 0$) for $\tau > 0$ ($< 0$). We will apply the rule to both J and W in our linearized network, where the firing rates $g_u(u_i)$ and $g_v(v_i)$ vary linearly with $u_i$ and $v_i$, so we will use Eqn. (6) with $x_j = u_j$ and $y_i = u_i$ or $v_i$ (measured from the fixed point $\bar{v}_i$), respectively.

We assume oscillatory input $\delta \mathbf{I} = \boldsymbol{\xi}^0 e^{-i\omega_0 t} + $ c.c. during learning. In the brain structures we are modeling, cholinergic modulation makes the long-range connections ineffective during learning [7]. Thus we set J = W = 0 in Eqn. (3) and find

$$u_i^+ = \frac{(\omega_0 + i\alpha)\xi_i^0 e^{-i\omega_0 t}}{2\alpha\omega_0 + i(\alpha^2 + \beta\gamma - \omega_0^2)} \equiv U_0 \xi_i^0 e^{-i\omega_0 t} \qquad (7)$$

and, from $(\partial_t + \alpha)v_i = \gamma u_i$,

$$v_i^+ = \frac{\gamma}{-i\omega_0 + \alpha} U_0 \xi_i^0 e^{-i\omega_0 t}. \qquad (8)$$

Using these in the learning rule (6) leads to

$$J_{ij} = 2J_0 \text{Re}\left[\tilde{A}(\omega_0)\xi_i^0 \xi_j^{0*}\right], \qquad W_{ij} = 2(\eta_W/\eta_J)J_0\gamma\text{Re}\left[\frac{\tilde{A}(\omega_0)\xi_i^0\xi_j^{0*}}{\alpha - i\omega_0}\right], \qquad (9)$$

where $\tilde{A}(\omega) = \int_{-\infty}^{\infty} d\tau \, A(\tau)e^{-i\omega\tau}$ is the Fourier transform of $A(\tau)$, $J_0 = 2\pi\eta_J|U_0|^2/\omega_0$, and $\eta_{J(W)}$ are the respective learning rates. When the rates are tuned such that $\eta_J = \eta_W\gamma\beta/(\alpha^2 + \omega_0^2)$ and when $\omega = \omega_0$, we have $M_{ij}^+ = J_0\tilde{A}(\omega_0)\xi_i^0\xi_j^{0*}$, a

generalization of the outer-product learning rule to the complex patterns $\boldsymbol{\xi}^\mu$ from the Hopfield-Hebb form for real-valued patterns. For learning multiple patterns $\boldsymbol{\xi}^\mu$, $\mu = 1, 2, ...$, the learned weights are simply sums of contributions from individual patterns like Eqns. (9) with $\xi_i^0$ replaced by $\xi_i^\mu$.

## 2.2  Recall phase

We return to the single-pattern problem and study the simple case when $\eta_J = \eta_W \gamma \beta / (\alpha^2 + \omega_0^2)$. Consider first an input pattern $\delta \mathbf{I} = \boldsymbol{\xi} e^{-\mathrm{i}\omega t} + \text{c.c.}$ that matches the stored pattern exactly ($\boldsymbol{\xi} = \boldsymbol{\xi}^0$), but possibly oscillating at a different frequency. We then find, using Eqns. (9) in Eqn. (3), the (positive-frequency) response

$$\mathbf{u}^+ = \frac{(\omega + \mathrm{i}\alpha)\boldsymbol{\xi}^0 e^{-\mathrm{i}\omega t}}{2\alpha\omega - \frac{J_0}{2}(\omega + \omega_0)\tilde{A}'(\omega_0) + \mathrm{i}[\alpha^2 + \beta\gamma - \frac{J_0}{2}(\omega + \omega_0)\tilde{A}''(\omega_0) - \omega^2]}. \tag{10}$$

where $\tilde{A}'(\omega_0) \equiv \mathrm{Re}\,\tilde{A}(\omega_0)$ and $\tilde{A}''(\omega_0) \equiv \mathrm{Im}\,\tilde{A}(\omega_0)$. For strong response at $\omega = \omega_0$, we require

$$\omega_0 = \sqrt{\alpha^2 + \beta\gamma - J_0\omega_0\tilde{A}''(\omega_0)}, \qquad J_0\tilde{A}'(\omega_0) \approx 2\alpha. \tag{11}$$

This means (1) the resonance frequency $\omega_0$ is determined by $\tilde{A}''$, (2) the effective damping $2\alpha - J_0\tilde{A}'$ should be small, and (3) deviation of $\omega$ from $\omega_0$ reduces the responses.

It is instructive to consider the case where the width of the time window for synaptic change is small compared with the oscillation period. Then we can expand $\tilde{A}(\omega_0)$ in $\omega_0$:

$$\tilde{A}'(\omega_0) \approx \int \mathrm{d}\tau A(\tau) \equiv a_0, \qquad \tilde{A}''(\omega_0) \approx -\omega_0 \int \mathrm{d}\tau \tau A(\tau) \equiv -\omega_0 a_1. \tag{12}$$

In particular, $A(\tau) = \delta(\tau)$ gives $a_0 = 1$ and $a_1 = 0$ and the conventional Hebbian learning [5]. Experimentally, $a_1 > 0$, implying a resonant frequency greater than the intrinsic local frequency, $\sqrt{\alpha^2 + \beta\gamma}$ obtained in the absence of long-range coupling.

If the drive $\boldsymbol{\xi}$ does not match the stored pattern (in phase and amplitude), the response will consist of two terms. The first has the form of Eqn. (10) but reduced in amplitude by an overlap factor $\boldsymbol{\xi}^{0*} \cdot \boldsymbol{\xi}$. (For convenience we use normalized pattern vectors.) The second term is proportional to the part of $\boldsymbol{\xi}$ orthogonal to the stored pattern. The J and W matrices do not act in this subspace, so the frequency dependence of this term is just that of uncoupled oscillators, i.e., Eqn. (10) with $J_0$ set equal to zero. This response is always highly damped and therefore small.

It is straightforward to extend this analysis to multiple imprinted patterns. The response consists of a sum of terms, one for each stored pattern. The term for each stored pattern is just like that just described in the single-stored-pattern case: it has one part for the input component parallel to the stored pattern and another part for the component orthogonal to the stored pattern.

We note that, in this linear analysis, an input which overlaps several stored patterns will (if the imprinting and input frequencies match) evoke a resonant response which is a linear combination of the stored patterns. Thus, a network tuned to operate in a nearly linear regime is able to interpolate in forming its representation of the input. For categorical associative memory, on the other hand, a network has to work in the extreme nonlinear limit, responding with only the strongest stored pattern in an input mixture. As our network operates near the threshold for spontaneous oscillations, we expect that it should exhibit properties intermediate between

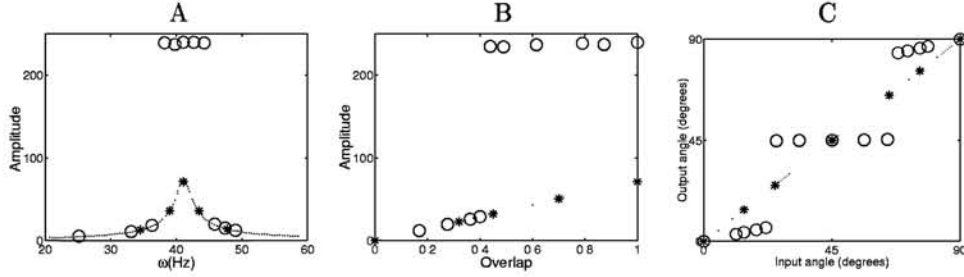

Figure 2: Circles show non-linear simulation results, stars show the linear simulation results, while the dotted line show the analytical prediction for the linearized model. A. Importance of frequency match: amplitude of response of output units as a function of the frequency of the current input. The frequency of the imprinted pattern is 41 Hz. B.Importance of amplitude and phase mismatch: amplitude of response as a function of overlap between current input and imprinted pattern (i.e. $|\boldsymbol{\xi}^{0*} \cdot \boldsymbol{\xi}|$), for different presented input patterns $\boldsymbol{\xi}$. C: Input - output relationship when two orthogonal patterns $\boldsymbol{\xi}^1$ and $\boldsymbol{\xi}^2$, have been imprinted at the same frequency $\omega = 41Hz$. The angle of input pattern with respect to $\boldsymbol{\xi}^1$ is shown as a function of the angle of the output pattern with respect to $\boldsymbol{\xi}^1$, for many different input patterns.

these limits. We find that this is indeed the case in the simulations reported in the next section. From our analysis it turns out that the network behaves like a Hopfield-memory (separate basins, without interpolation capability) for patterns with different imprinting frequencies, but at the same time it is able to interpolate among patterns which share a common frequency.

## 3 Simulations

Checking the validity of our linear approximation in the analysis, we performed numerical simulations of both the non-linear equations (1,2) and the linearized ones (3). We simulated the recall phase of a network consisting of 10 excitatory and 10 inhibitory cells. The connections $J_{ij}$ and $W_{ij}$ were calculated from Eqns. (9), where we used the approximations (12) for the kernel shape $A(\tau)$. Parameters were set in such a way that the selective resonance was in the 40-Hz range. In non-linear simulations we used different piecewise linear activation functions for $g_u()$ and $g_v()$, as shown in Fig.1B. We chose the parameters of the functions $g_u()$ and $g_v()$ so that the network equilibrium points $\bar{u}_i, \bar{v}_i$ were close to, but below, the high-gain region, i.e. at the points marked with crosses in Fig. 1B.

The results confirm that when the input pattern matches the imprinted one in frequency, amplitude and phase, the network responds with strong resonant oscillations. However, it does not resonate if the frequencies do not match, as shown in the frequency tuning curve in Fig. 2A. The behavior when the two frequencies are close to each other differs in the linear and nonlinear cases. However, in both cases a sharp selectivity in frequency is observed. The dependence on the overlap between the input and the stored pattern is shown in Fig. 2B. The non-linear case, indicated by circles, should be compared with the linear case, where the amplitude is always linear in the overlap. In the nonlinear case, the linearity holds roughly only for overlaps lower than about 0.4; for larger overlaps the amplification is as high as for the perfect match case. This means that input patterns with an overlap with the imprinted one greater than 0.4 lie within the basis of attraction of the

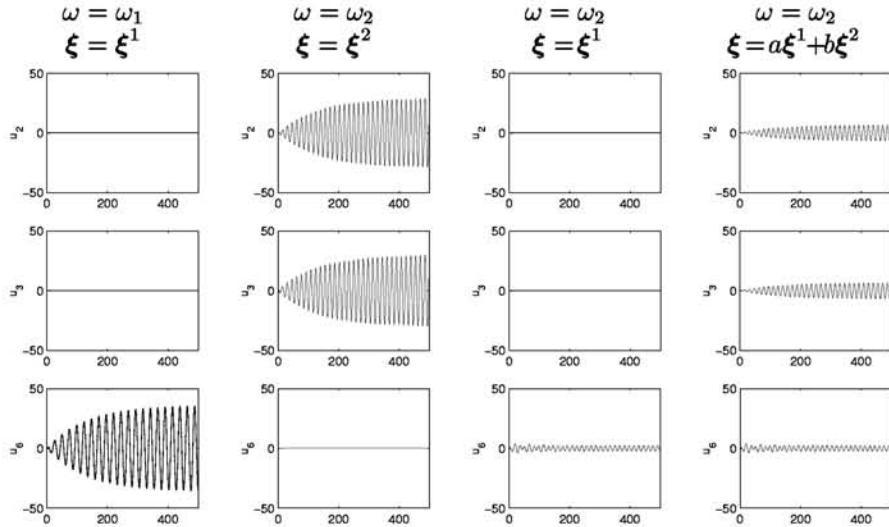

Figure 3: Frequency selectivity: Response evoked on 3 of the 10 neurons. Oscillatory patterns $\boldsymbol{\xi}^1 e^{-i\omega_1 t} +$ c.c. and $\boldsymbol{\xi}^2 e^{-i\omega_2 t}$c.c. have been imprinted, with $\boldsymbol{\xi}^1 \perp \boldsymbol{\xi}^2$ and $\omega_1 = 41$ Hz, $\omega_2 = 63$ Hz. During the learning phases the parameter $a_1$ of kernel was tuned appropriately, i.e. $a_1 = 0.1$ when imprinting $\boldsymbol{\xi}^1$ and $a_1 = 1.1$ when imprinting $\boldsymbol{\xi}^2$.

imprinted pattern.

The response elicited when two orthogonal patterns have been imprinted with the same frequency is shown in Fig. 2C. Let $\boldsymbol{\xi}^1 e^{-i\omega_o t} +$ c.c. and $\boldsymbol{\xi}^2 e^{-i\omega_o t} +$ c.c. denote the imprinted patterns, and $\boldsymbol{\xi} e^{-i\omega_o t} +$ c.c. be the input to the trained network. In both linear and non-linear simulations the network responds vigorously(with high-amplitude oscillations) to the drive if $\boldsymbol{\xi}$ is in the subspace spanned by the imprinted patterns, and fails to respond appreciably if $\boldsymbol{\xi}$ is orthogonal to that plane. When the input pattern $\boldsymbol{\xi}$ is in the plane spanned by the stored patterns, the resonant response $\mathbf{u}$ also lies in this plane. However, while in the linear case the output is proportional to the input, in agreement with the analytical analysis, in the non-linear case there are preferred directions, in the stored pattern plane. The figure shows that, in case simulated here, there are three stable attractors: $\boldsymbol{\xi}^1$, $\boldsymbol{\xi}^2$, and the symmetric linear combination $(\boldsymbol{\xi}^1 + \boldsymbol{\xi}^2)/\sqrt{2}$.

Finally we performed linear simulations storing two orthogonal patterns $\boldsymbol{\xi}^1 e^{-i\omega_1 t} +$ c.c. and $\boldsymbol{\xi}^2 e^{-i\omega_2 t} +$ c.c. with two different imprinting frequencies. Fig. 3 shows a good performance of the network in separating the basins of attraction in this case. The response to a linear combination of the two patterns, $(a\boldsymbol{\xi}^1 + b\boldsymbol{\xi}^2)e^{-i\omega_2 t} +$ c.c. is proportional to the part of the input whose imprinting frequency matches the current driving frequency. Linear combinations of the two imprinted patterns are not attractors if the two patterns do not share the same imprinting frequency.

## 4  Summary and Discussion

We have presented a model of learning for memory or input representations in neural networks with input-driven oscillatory activity. The model structure is an abstrac-

tion of the hippocampus or the olfactory cortex. We propose a simple generalized Hebbian rule, using temporal-activity-dependent LTP and LTD, to encode both magnitudes and phases of oscillatory patterns into the synapses in the network. After learning, the model responds resonantly to inputs which have been learned (or, for networks which operate essentially linearly, to linear combinations of learned inputs), but negligibly to other input patterns. Encoding both amplitude and phase enhances computational capacity, for which the price is having to learn both the excitatory-to-excitatory and the excitatory-to-inhibitory connections. Our model puts contraints on the form of the learning kernal $A(\tau)$ that should be experimenally observed, e.g., for small oscillation frequencies, it requires that the overall LTP dominates the overall LTD, but this requirement should be modified if the stored oscillations are of high frequencies. Plasticity in the excitatory-to-inhibitory connections (for which experimental evidence and investigation is still scarce) is required by our model for storing phase locked but unsynchronous oscillation patterns.

As for the Hopfield model, we distinguish two functional phases: (1) the learning phase, in which the system is clamped dynamically to the external inputs and (2) the recall phase, in which the system dynamics is determined by both the external inputs and the internal interactions.

A special property of our model in the linear regime is the following interpolation capability: under a given oscillation frequency, once the system has learned a set of representation states, all other states in the subspace spanned by the learned states can also evoke vigorous responses. Hippocampal place cells could employ such a representation. Each cell has a localised "place field", and the superposition of activity of several cells wth nearby place fields can represent continuously-varying position. The locality of the place fields also means that this representation is conservative (and thus robust), in the sense that interpolation does not extend beyond the spatial range of the experienced locations or to locations in between two learned but distant and disjoint spatial regions.

Of course, this interpolation property is not always desirable. For instance, in categorical memory, one does not want inputs which are linear combinations of stored patterns to elicit responses which are also similar linear combinations. Suitable nonlinearity can (as we saw in the last section), enable the system to perform categorization: one way involved storing different patterns (or, by implication, different classes of patterns) at different frequencies. For instance, in a multimodal area, "place fields" might be stored at one oscillation frequency, and (say) odor memories at another. It seems likely to us that the brain may employ different kinds and degrees of nonlinearity in different areas or at different times to enhance the versatility of its computations.

### References

[1] H Markram, J Lubke, M Frotscher, and B Sakmann, *Science* **275** 213 (1997).

[2] J C Magee and D Johnston, *Science* **275** 209 (1997).

[3] D Debanne, B H Gahwiler, and S M Thompson, *J Physiol* **507** 237 (1998).

[4] G Q Bi and M M Poo, *J Neurosci* **18** 10464 (1998).

[5] Z Li and J Hertz, *Network: Computation in Neural Systems* **11** 83-102 (2000).

[6] Z Li and J J Hopfield, *Biol Cybern* **61** 379-92 (1989).

[7] M E Hasselmo, *Neural Comp* **5** 32-44 (1993).
